# The "Softmax" Nonlinearity: Derivation Using Statistical Mechanics and Useful Properties as a Multiterminal Analog Circuit Element

**I. M. Elfadel**
Research Laboratory of Electronics
Massachusetts Institute of Technology
Cambridge, MA 02139

**J. L. Wyatt, Jr.**
Research Laboratory of Electronics
Massachusetts Institute of Technology
Cambridge, MA 02139

## Abstract

We use mean-field theory methods from Statistical Mechanics to derive the "softmax" nonlinearity from the discontinuous winner-take-all (WTA) mapping. We give two simple ways of implementing "softmax" as a multiterminal network element. One of these has a number of important network-theoretic properties. It is a reciprocal, passive, incrementally passive, nonlinear, resistive multiterminal element with a content function having the form of information-theoretic entropy. These properties should enable one to use this element in nonlinear RC networks with such other reciprocal elements as resistive fuses and constraint boxes to implement very high speed analog optimization algorithms using a minimum of hardware.

## 1 Introduction

In order to efficiently implement nonlinear optimization algorithms in analog VLSI hardware, maximum use should be made of the natural properties of the silicon medium. Reciprocal circuit elements facilitate such an implementation since they

can be combined with other reciprocal elements to form an analog network having Lyapunov-like functions: the network content or co-content. In this paper, we show a reciprocal implementation of the "softmax" nonlinearity that is usually used to enforce local competition between neurons [Peterson, 1989]. We show that the circuit is passive and incrementally passive, and we explicitly compute its content and co-content functions. This circuit adds a new element to the library of the analog circuit designer that can be combined with reciprocal constraint boxes [Harris, 1988] and nonlinear resistive fuses [Harris, 1989] to form fast, analog VLSI optimization networks.

## 2   Derivation of the Softmax Nonlinearity

To a vector $\mathbf{y} \in \Re^n$ of distinct real numbers, the discrete winner-take-all (WTA) mapping $\mathbf{W}$ assigns a vector of binary numbers by giving the value 1 to the component of $\mathbf{y}$ corresponding to $\max_{1 \leq i \leq n} y_i$ and the value 0 to the remaining components. Formally, $\mathbf{W}$ is defined as

$$\mathbf{W}(\mathbf{y}) = (W_1(\mathbf{y}), \ldots, W_n(\mathbf{y}))^T$$

where for every $1 \leq j \leq n$,

$$W_j(\mathbf{y}) = \begin{cases} 1 & \text{if } y_j > y_i, \ \forall \ 1 \leq i \leq n \\ 0 & \text{otherwise} \end{cases} \tag{1}$$

Following [Geiger, 1991], we assign to the vector $\mathbf{y}$ the "energy" function

$$E_{\mathbf{y}}(\mathbf{z}) = -\sum_{k=1}^{n} z_k y_k = -\mathbf{z}^T \mathbf{y}, \ \mathbf{z} \in \mathcal{V}_n, \tag{2}$$

where $\mathcal{V}_n$ is the set of vertices of the unit simplex $\mathcal{S}_n = \{\mathbf{x} \in \Re^n, x_i \geq 0, 1 \leq i \leq n \text{ and } \sum_{k=1}^{n} x_k = 1\}$. Every vertex in the simplex encodes one possible winner. It is then easy to show that $\mathbf{W}(\mathbf{y})$ is the solution to the linear programming problem

$$\max_{\mathbf{z} \in \mathcal{V}_n} \sum_{k=1}^{n} z_k y_k.$$

Moreover, we can assign to the energy $E_{\mathbf{y}}(\mathbf{z})$ the Gibbs distribution

$$P_{\mathbf{y}}(\mathbf{z}) = P_{\mathbf{y}}(z_1, \ldots, z_n) = \frac{e^{-E_{\mathbf{y}}(\mathbf{z})/T}}{Z_T}$$

where $T$ is the temperature of the heat bath and $Z_T$ is a normalizing constant. Then one can show that the mean of $z_j$ considered as a random variable is given by [Geiger, 1991]

$$F_j(\mathbf{y}/T) \stackrel{\Delta}{=} \overline{z_j} = \frac{e^{y_j/T}}{Z_T} = \frac{e^{y_j/T}}{\sum_{i=1}^{n} e^{y_i/T}}.$$

The mapping $\mathbf{F} : \Re^n \rightarrow \Re^n$ whose components are the $F_j$'s, $1 \leq j \leq n$, is the *generalized sigmoid mapping* [Peterson, 1989] or "softmax". It plays, in WTA networks, a role similar to that of the sigmoidal function in Hopfield and backpropagation

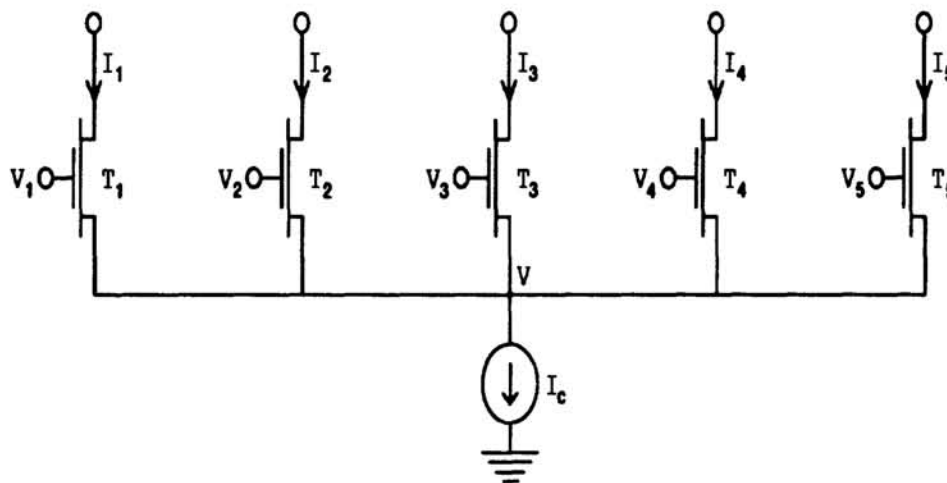

Figure 1: A circuit implementation of softmax with 5 inputs and 5 outputs. This circuit is operated in subthreshold mode, takes the gates voltages as inputs and gives the drain currents as outputs. This circuit is *not* a reciprocal multiterminal element.

networks [Hopfield, 1984, Rumelhart, 1986] and is usually used for enforcing competitive behavior among the neurons of a single cluster in networks of interacting clusters [Peterson, 1989, Waugh, 1993].

For $\mathbf{y} \in \Re^n$, we denote by $\mathbf{F}_T(\mathbf{y}) \triangleq \mathbf{F}(\mathbf{y}/T)$. The softmax mapping satisfies the following properties:

1. The mapping $F_T$ converges pointwise to $\mathbf{W}$ over $\Re^n$ as $T \to 0$ and to the center of mass of $\mathcal{S}_n$, $\frac{1}{n}\mathbf{e} = \frac{1}{n}(1, 1, \ldots, 1)^T \in \Re^n$, as $T \to +\infty$.

2. The Jacobian $D\mathbf{F}$ of the softmax mapping is a symmetric $n \times n$ matrix that satisfies

$$DF(\mathbf{y}) = diag\,(F_k(\mathbf{y})) - \mathbf{F}(\mathbf{y})\mathbf{F}(\mathbf{y})^T. \qquad (3)$$

It is always singular with the vector e being the only eigenvector corresponding to the zero eigenvalue. Moreover, all its eigenvalues are upper-bounded by $\max_{1 \le k \le n} F_k(\mathbf{y}) < 1$.

3. The softmax mapping is a gradient map, i.e, there exists a "potential" function $\mathcal{P} : \Re^n \to \Re$ such that $\mathbf{F} = \nabla\mathcal{P}$. Moreover $\mathcal{P}$ is convex.

The symbol $\mathcal{P}$ was chosen to indicate that it is a *potential* function. It should be noted that if $\mathbf{F}$ is the gradient map of $\mathcal{P}$ then $\mathbf{F}_T$ is the gradient map of $T\mathcal{P}_T$ where $\mathcal{P}_T(\mathbf{y}) = \mathcal{P}(\mathbf{y}/T)$. In a related paper [Elfadel, 1993], we have found that the convexity of $\mathcal{P}$ is essential in the study of the global dynamics of analog WTA networks. Another instance where the convexity of $\mathcal{P}$ was found important is the one reported in [Kosowsky, 1991] where a mean-field algorithm was proposed to solve the linear assignment problem.

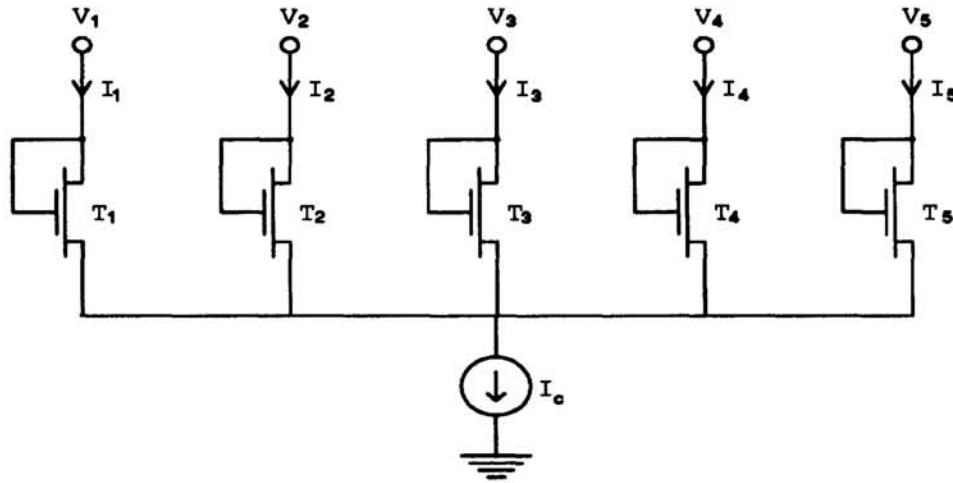

Figure 2: Modified circuit implementation of softmax. In this circuit all the transistors are diode-connected, and all the drain currents are well in saturation region. Note that for every transistor, both the voltage input and the current output are on the same wire – the drain. This circuit is a reciprocal multiterminal element.

## 3    Circuit Implementations and Properties

Now we propose two simple CMOS circuit implementations of the generalized sigmoid mapping. See Figures 1 and 2. When the transistors are operated in the subthreshold region the drain currents $i_1, \ldots, i_n$ are the outputs of a softmax mapping whose inputs are the gate voltages $v_1, \ldots, v_n$. The explicit $v - i$ characteristics are given by

$$i_m = I_c \frac{\exp(\kappa v_m / V_0)}{\sum_{p=1}^{n} \exp(\kappa v_p / V_0)}, \tag{4}$$

where $\kappa$ is a process-dependent parameter and $V_0$ is the thermal voltage ([Mead, 1989],p. 36). These circuits have the interesting properties of being unclocked and parallel. Moreover, the competition constraint is imposed naturally through the KCL equation and the control current source. From a complexity point of view, this circuit is most striking since it computes $n$ exponentials, $n$ ratios, and $n - 1$ sums in one time constant! A derivation similar to the above was independently carried out in [Waugh, 1993] for the circuit of Figure 1. Although the first circuit implements softmax, it has two shortcomings. The first is practical: the separation between inputs and outputs implies additional wiring. The second is theoretical: this circuit is *not* a reciprocal multiterminal element, and therefore it can't be combined with other reciprocal elements like resistive fuses or constraint boxes to design analog, reciprocal optimization networks.

Therefore, we only consider the circuit of Figure 2 and let **v** and **i** be the $n$-dimensional vectors representing the input voltages and the output currents, respectively. [1] The softmax mapping $\mathbf{i} = \mathbf{F}(\mathbf{v})$ represents a voltage-controlled, nonlinear,

resistive multiterminal element. The main result of our paper is the following:[2]

**Theorem 1** *The softmax multiterminal element* **F** *is reciprocal, passive, locally passive and has a co-content function given by*

$$\Phi(\mathbf{v}) = \frac{1}{\kappa} I_c V_0 \ln \sum_{m=1}^{n} \exp(\kappa v_m / V_0) \qquad (5)$$

*and a content function given by*

$$\Phi^*(\mathbf{i}) = \frac{I_c V_0}{\kappa} \sum_{m=1}^{n} \frac{i_m}{I_c} \ln \frac{i_m}{I_c}. \qquad (6)$$

Thus, with this reciprocal, locally passive implementation of the softmax mapping, we have added a new circuit element to the library of the circuit designer. Note that this circuit element implements in an analog way the constraint $\sum_{k=1}^{n} y_k = 1$ defining the unit simplex $S_n$. Therefore, it can be considered a *nonlinear* constraint box [Harris, 1988] that can be used in reciprocal networks to implement analog optimization algorithms.

The expression of $\Phi^*$ is a strong reminder of the information-theoretic definition of entropy. We suggest the name "entropic resistor" for the circuit of Figure 2.

## 4   Conclusions

In this paper, we have discussed another instance of convergence between the statistical physics paradigm of Gibbs distributions and analog circuit implementation in the context of the winner-take-all function. The problem of using the simple, reciprocal circuit implementation of softmax to design analog networks for finding near optimal solutions of the linear assignment problem [Kosowsky, 1991] or the quadratic assignment problem [Simic, 1991] is still open and should prove a challenging task for analog circuit designers.

**Acknowledgements**

I. M. Elfadel would like to thank Alan Yuille for many helpful discussions and Fred Waugh for helpful discussions and for communicating the preprint of [Waugh, 1993]. This work was supported by the National Science Foundation under Grant No. MIP-91-17724.

## Footnotes

[1] Compare with Lazarro et. al.'s WTA circuit [Lazzaro, 1989] whose inputs are currents and outputs are voltages.

[2]The concepts of reciprocity, passivity, content, and co-content are fundamental to nonlinear circuit theory. They are carefully developed in [Wyatt, 1992].

## References

[Peterson, 1989] C. Peterson and B. Söderberg. A new method for mapping optimization problems onto neural networks. *International Journal of Neural Systems*, 1(1):3 − 22, 1989.

[Harris, 1988]    J. G. Harris. Solving early vision problems with VLSI constraint networks. In *Neural Architectures for Computer Vision Workshop, AAAI-88*, Minneapolis, MN, 1988.

[Harris, 1989]    J. G. Harris, C. Koch, J. Luo, and J. Wyatt. Resistive fuses: Analog hardware for detecting discontinuities in early vision. In C. Mead and M. Ismail, editors, *Analog VLSI Implemenation of Neural Systems*. Kluwer Academic Publishers, 1989.

[Geiger, 1991]    D. Geiger and A. Yuille. A common framework for image segmentation. *Int. J. Computer Vision*, 6:227 – 253, 1991.

[Hopfield, 1984]  J. J. Hopfield. Neurons with graded responses have collective computational properties like those of two-state neurons. *Proc. Nat'l Acad. Sci., USA*, 81:3088–3092, 1984.

[Rumelhart, 1986] D. E. Rumelhart *et. al.* *Parallel Distributed Processing*, volume 1. MIT Press, 1986.

[Waugh, 1993]    F. R. Waugh and R. M. Westervelt. Analog neural networks with local competition. I. dynamics and stability. *Physical Review E*, 1993. in press.

[Elfadel, 1993]    I. M. Elfadel. Global dynamics of winner-take-all networks. In *SPIE Proceedings, Stochastic and Neural Methods in Image Processing*, volume 2032, pages 127 – 137, San Diego, CA, 1993.

[Kosowsky, 1991] J. J. Kosowsky and A. L. Yuille. The invisible hand algorithm: Solving the assignment problem with statistical physics. TR # 91-1, Harvard Robotics Laboratory, 1991.

[Mead, 1989]     Carver Mead. *Analog VLSI and Neural Systems*. Addison-Wesley, 1989.

[Lazzaro, 1989]   J. Lazarro, S. Ryckebush, M. Mahowald, and C. Mead. Winner-take-all circuits of $O(n)$ complexity. In D. S. Touretsky, editor, *Advances in Neural Information Processing Systems I*, pages 703 – 711. Morgan Kaufman, 1989.

[Wyatt, 1992]    J. L. Wyatt. *Lectures on Nonlinear Circuit Theory*. MIT VLSI memo # 92-685, 1992.

[Simic, 1991]    P. D. Simic. Constrained nets for graph matching and other quadratic assignment problems. *Neural Computation*, 3:169 – 281, 1991.
